# Reinforcement Learning Predicts the Site of Plasticity for Auditory Remapping in the Barn Owl

**Alexandre Pouget†**
alex@salk.edu

**Cedric Deffayet‡**
cedric@salk.edu

**Terrence J. Sejnowski†**
terry@salk.edu

†Howard Hughes Medical Institute
The Salk Institute
La Jolla, CA 92037
Department of Biology
University of California, San Diego
and
‡Ecole Normale Superieure
45 rue d'Ulm
75005 Paris, France

## Abstract

The auditory system of the barn owl contains several spatial maps. In young barn owls raised with optical prisms over their eyes, these auditory maps are shifted to stay in register with the visual map, suggesting that the visual input imposes a frame of reference on the auditory maps. However, the optic tectum, the first site of convergence of visual with auditory information, is not the site of plasticity for the shift of the auditory maps; the plasticity occurs instead in the inferior colliculus, which contains an auditory map and projects into the optic tectum. We explored a model of the owl remapping in which a global reinforcement signal whose delivery is controlled by visual foveation. A hebb learning rule gated by reinforcement learned to appropriately adjust auditory maps. In addition, reinforcement learning preferentially adjusted the weights in the inferior colliculus, as in the owl brain, even though the weights were allowed to change throughout the auditory system. This observation raises the possibility that the site of learning does not have to be genetically specified, but could be determined by how the learning procedure interacts with the network architecture.

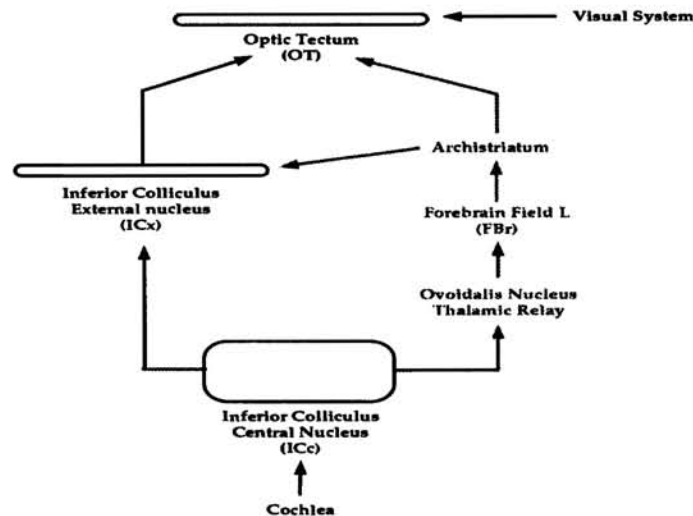

Figure 1: Schematic view of the auditory pathways in the barn owl.

# 1   Introduction

The barn owl relies primarily on sounds to localize prey [6] with an accuracy vastly superior to that of humans. Figure 1A illustrates some of the nuclei involved in processing auditory signals. The barn owl determines the location of sound sources by comparing the time and amplitude differences of the sound wave between the two ears. These two cues are combined together for the first time in the shell and core of the inferior colliculus (ICc) which is shown at the bottom of the diagram. Cells in the ICc are frequency tuned and subject to spatial aliasing. This prevents them from unambiguously encoding the position of objects. The first unambiguous auditory map is found at the next stage, in the external capsule of the inferior colliculus (ICx) which itself projects to the optic tectum (OT). The OT is the first subforebrain structure which contains a multimodal spatial map in which cells typically have spatially congruent visual and auditory receptive fields.

In addition, these subforebrain auditory pathways send one major collateral toward the forebrain via a thalamic relay. These collaterals originate in the ICc and are thought to convey the spatial location of objects to the forebrain [3]. Within the forebrain, two major structures have been involved in auditory processing: the archistriatum and field L. The archistriatum sends a projection to both the inferior colliculus and the optic tectum.

Knudsen and Knudsen (1985) have shown that these auditory maps can adapt to systematic changes in the sensory input. Furthermore, the adaptation appears to be under the control of visual input, which imposes a frame of reference on the incoming auditory signals. In owls raised with optical prisms, which introduce a systematic shift in part of the visual field, the visual map in the optic tectum was identical to that found in control animals, but the auditory map in the ICx was shifted by the amount of visual shift introduced by the prisms. This plasticity ensures that the visual and auditory maps stay in spatial register during growth

and other perturbations to sensory mismatch.

Since vision instructs audition, one might expect the auditory map to shift in the optic tectum, the first site of visual-auditory convergence. Surprisingly, Brainard and Knudsen (1993b) observed that the synaptic changes took place between the ICc and the ICx, one synapse before the site of convergence.

These observations raise two important questions: First, how does the animal knows how to adapt the weights in the ICx in the absence of a visual teaching signal? Second, why does the change take place at this particular location and not in the OT where a teaching signal would be readily available?

In a previous model [7], this shift was simulated using backpropagation to broadcast the error back through the layers and by constraining the weights changes to the projection from the ICc to ICx. There is, however, no evidence for a feedback projection between from the OT to the ICx that could transmit the error signal; nor is there evidence to exclude plasticity at other synapses in these pathways.

In this paper, we suggest an alternative approach in which vision guides the remapping of auditory maps by controlling the delivery of a scalar reinforcement signal. This learning proceeds by generating random actions and increasing the probability of actions that are consistently reinforced [1, 5]. In addition, we show that reinforcement learning correctly predicts the site of learning in the barn owl, namely at the ICx-ICc synapse, whereas backpropagation [8] does not favor this location when plasticity is allowed at every synapse. This raises a general issue: the site of synaptic adjustment might be imposed by the combination of the architecture and learning rule, without having to restrict plasticity to a particular synapse.

## 2  Methods

### 2.1  Network Architecture

The network architecture of the model based on the barn owl auditory system, shown in figure 2A, contains two parallel pathways. The input layer was an 8x21 map corresponding to the ICc in which units responded to frequency and interaural phase differences. These responses were pooled together to create auditory spatial maps at subsequent stages in both pathways. The rest of the network contained a series of similar auditory maps, which were connected topographically by receptive fields 13 units wide. We did not distinguish between field L and the archistriatum in the forebrain pathways and simply used two auditory maps, both called FBr.

We used multiplicative (sigma-pi) units in the OT whose activities were determined according to:

$$y_i = \sum_j w_{ij}^{FBr} y_j^{FBr} w_{jk}^{FBr} y_j^{ICx} \qquad (1)$$

The multiplicative interaction between ICx and FBr activities was an important assumption of our model. It forced the ICx and FBr to agree on a particular position before the OT was activated. As a result, if the ICx-OT synapses were modified during learning, the ICx-FBr synapses had to be changed accordingly.

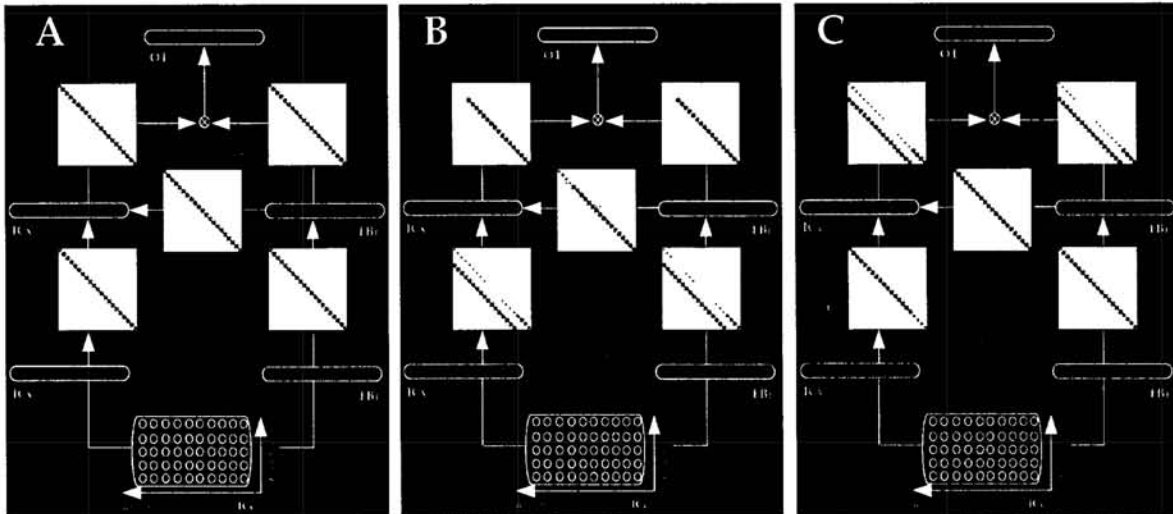

Figure 2: Schematic diagram of weights (white blocks) in the barn owl auditory system. A) Diagram of the initial weights in the network. B) Pattern of weights after training with reinforcement learning on a prism-induced shift of four units. The remapping took place within the ICx and FBr. C) Pattern of weights after training with backpropagation. This time the ICx-OT and FBr-OT weights changed.

Weights were clipped between 5.0 and 0.01, except for the FBr-ICx connections whose values were allowed to vary between 8.0 and 0.01. The minimum values were set to 0.01 instead of zero to prevent getting trapped in unstable local minima which are often associated with weights values of zero. The strong coupling between FBr and ICx was another important assumption of the model whose consequence will be discussed in the last section.

Examples were generated by simply activating one unit in the ICc while keeping the others to zero, thereby simulating the pattern of activity that would be triggered by a single localized auditory stimulus. In all simulations, we modeled a prism-induced shift of four units.

## 2.2   Reinforcement learning

We used stochastic units and trained the network using reinforcement learning [1]. The weighted sum of the inputs, $net_i$, passed through a sigmoid, f(x), is interpreted as the probability, $p_i$, that the unit will be active:

$$p_i = f(net_i) * 0.99 + 0.01 \qquad (2)$$

were the output of the unit $y_i$ was:

$$y_i = \begin{cases} 0 & \text{with probability } 1 - p_i \\ 1 & \text{with probability } p_i \end{cases} \qquad (3)$$

Because of the form of the equation for $p_i$, all units in the network had a small probability (0.01) of being spontaneously active in the absence of any inputs. This is what allowed the network to perform a stochastic search in action space to find which actions were consistently associated with positive reinforcement.

We ensured that at most one unit was active per trial by using a winner-take-all competition in each layer.

Adjustable weights in the network were updated after each training examples with hebb-like rule gated by reinforcement:

$$\Delta w_{ij} = \epsilon y_i \frac{\partial net_i}{\partial w_{ij}} r \qquad (4)$$

A trial consisted in choosing a random target location for auditory input (ICc) and the output of the OT was used to generate a head movement. The reinforcement, $r$, was then set to 1 for head movements resulting in the foveation of the stimulus and to -0.05 otherwise.

## 2.3 Backpropagation

For the backpropagation network, we used deterministic units with sigmoid activation functions in which the output of a unit was given by:

$$y_i = f(net_i) \qquad (5)$$

where $net_i$ is the weighted sum of the inputs as before.

The chain rule was used to compute the partial derivatives of the squared error, $E$, with respect to each weights and the weights were updated after each training example according to:

$$\Delta w_{ij} = -\epsilon \frac{\partial E}{\partial w_{ij}} \qquad (6)$$

The target vectors were similar to the input vectors, namely only one OT units was required to be activated for a given pattern, but at a position displaced by 4 units compared to the input.

## 3 Results

### 3.1 Learning site with reinforcement

In a first set of simulation we kept the ICc-ICx and ICc-FBr weights fixed. Plasticity was allowed at these site in later simulations.

Figure 2A shows the initial set of weights before learning starts. The central diagonal lines in the weight diagrams illustrate the fact that each unit receives only one non-zero weight from the unit in the layer below at the same location.

There are two solutions to the remapping: either the weights change within the ICx and FBr, or from the ICx and the FBr to the OT. As shown in figure 2B, reinforcement learning converged to the first solution. In contrast, the weights between the other layers were unaltered, even though they were allowed to change.

To prove that the network could have actually learned the second solution, we trained a network in which the ICc-ICx weights were kept fixed. As we expected, the network shifted its maps simultaneously in both sets of weights converging onto the OT, and the resulting weights were similar to the ones illustrated in figure 2C. However, to reach this solution, three times as many training examples were needed.

The reason why learning in the ICx and FBr were favored can be attributed to probabilistic nature of reinforcement learning. If the probability of finding one solution is $p$, the probability of finding it twice independently is $p^2$. Learning in the ICx and FBR is not independent because of the strong connection from the FBr to the ICx. When the remapping is learned in the FBR this connection automatically remapped the activities in the ICx which in turn allows the ICx-ICx weights to remap appropriately. In the OT on the other hand, the multiplicative connection between the ICx and FBr weights prevent a cooperation between this two sets of weights. Consequently, they have to change independently, a process which took much more training.

## 3.2   Learning at the ICc-ICx and ICc-FBr synapses

The aliasing and sharp frequency tuning in the response of ICc neurons greatly slows down learning at the ICc-ICx and ICc-FBr synapses. We found that when these synapses were free to change, the remapping still took place within the ICx or FBr (figure 3).

## 3.3   Learning site with backpropagation

In contrast to reinforcement learning, backpropagation adjusted the weights in two locations: between the ICx and the OT and between the Fbr and OT (figure 2C). This is the consequence of the tendency of the backpropagation algorithm to first change the weights closest to where the error is injected.

## 3.4   Temporal evolution of weights

Whether we used reinforcement or supervised learning, the map shifted in a very similar way. There was a simultaneous decrease of the original set of weights with a simultaneous increase of the new weights, such that both sets of weights coexisted half way through learning. This indicates that the map shifted directly from the original setting to the new configuration without going through intermediate shifts.

This temporal evolution of the weights is consistent the findings of Brainard and Knudsen (1993a) who found that during the intermediate phase of the remapping, cells in the inferior colliculus typically have two receptive fields. More recent work however indicates that for some cells the remapping is more continuous(Brainard and Knudsen, personal communication), a behavior that was not reproduced by either of the learning rule.

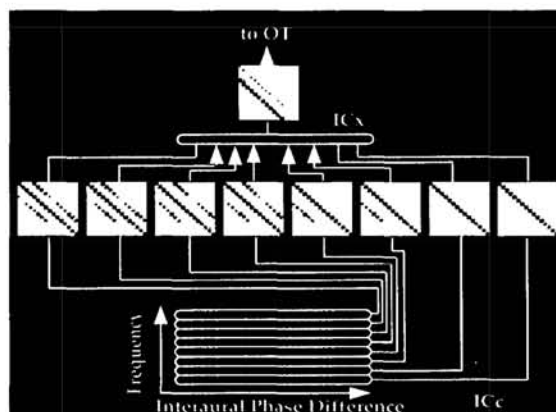

Figure 3: Even when the ICc-ICx weights are free to change, the network update the weights in the ICx first. A separate weight matrix is shown for each isofrequency map from the ICc to ICx. The final weight matrices were predominantly diagonal; in contrast, the weight matrix in ICx was shifted.

## 4 Discussion

Our simulations suggest a biologically plausible mechanism by which vision can guide the remapping of auditory spatial maps in the owl's brain. Unlike previous approaches, which relied on visual signals as an explicit teacher in the optic tectum [7], our model uses a global reinforcement signal whose delivery is controlled by the foveal representation of the visual system. Other global reinforcement signals would work as well. For example, a part of the forebrain might compare auditory and visual patterns and report spatial mismatch between the two. This signal could be easily incorporated in our network and would also remap the auditory map in the inferior colliculus.

Our model demonstrates that the site of synaptic plasticity can be constrained by the interaction between reinforcement learning and the network architecture. Reinforcement learning converged to the most probably solution through stochastic search. In the network, the strong lateral coupling between ICx and FBr and the multiplicative interaction in the OT favored a solution in which the remapping took place simultaneously in the ICx and FBr. A similar mechanism may be at work in the barn owl's brain. Colaterals from FBr to ICx are known to exist, but the multiplicative interaction has not been reported in the barn owl optic tectum.

Learning mechanisms may also limit synaptic plasticity. NMDA receptors have been reported in the ICx, but they might not be expressed at other synapses. There may, however, be other mechanisms for plasticity.

The site of remapping in our model was somewhat different from the existing observations. We found that the change took place *within* the ICx whereas Brainard and Knudsen [3] report that it is *between* the ICc and the ICx. A close examination of their data (figure 11 in [3]) reveals that cells at the bottom of ICx were not

remapped, as predicted by our model, but at the same time, there is little anatomical or physiological evidence for a functional and hierarchical organization within the ICx. Additional recordings are need to resolve this issue. We conclude that for the barn owl's brain, as well as for our model, synaptic plasticity within ICx was favored over changes between ICc and ICx. This supports the hypothesis that reinforcement learning is used for remapping in the barn owl auditory system.

## Acknowledgments

We thank Eric Knudsen and Michael Brainard for helpful discussions on plasticity in the barn owl auditory system and the results of unpublished experiments. Peter Dayan and P. Read Montague helped with useful insights on the biological basis of reinforcement learning in the early stages of this project.

## References

[1] A.G. Barto and M.I. Jordan. Gradient following without backpropagation in layered networks. *Proc. IEEE Int. Conf. Neural Networks*, 2:629–636, 1987.

[2] M.S. Brainard and E.I. Knudsen. Dynamics of the visual calibration of the map of interaural time difference in the barn owl's optic tectum. In *Society For Neuroscience Abstracts*, volume 19, page 369.8, 1993.

[3] M.S. Brainard and E.I. Knudsen. Experience-dependent plasticity in the inferior colliculus: a site for visual calibration of the neural representation of auditory space in the barn owl. *The journal of Neuroscience*, 13:4589–4608, 1993.

[4] E. Knudsen and P. Knudsen. Vision guides the adjustment of auditory localization in the young barn owls. *Science*, 230:545–548, 1985.

[5] P.R. Montague, P. Dayan, S.J. Nowlan, A. Pouget, and T.J. Sejnowski. Using aperiodic reinforcement for directed self-organization during development. In S.J. Hanson, J.D. Cowan, and C.L. Giles, editors, *Advances in Neural Information Processing Systems*, volume 5. Morgan-Kaufmann, San Mateo, CA, 1993.

[6] R.S. Payne. Acoustic location of prey by barn owls (tyto alba). *Journal of Experimental Biology*, 54:535–573, 1970.

[7] D.J. Rosen, D.E. Rumelhart, and E.I. Knudsen. A connectionist model of the owl's sound localization system. In *Advances in Neural Information Processing Systems*, volume 6. Morgan-Kaufmann, San Mateo, CA, 1994.

[8] D.E. Rumelhart, G.E. Hinton, and R.J. Williams. Learning internal representations by error propagation. In D. E. Rumelhart, J. L. McClelland, and the PDP Research Group, editors, *Parallel Distributed Processing*, volume 1, chapter 8, pages 318–362. MIT Press, Cambridge, MA, 1986.